# Decoding of Neuronal Signals in Visual Pattern Recognition

**Emad N Eskandar**
Laboratory of Neuropsychology
National Institute of Mental Health
Bethesda MD 20892 USA

**Barry J Richmond**
Laboratory of Neuropsychology
National Institute of Mental Health
Bethesda MD 20892 USA

**John A Hertz**
NORDITA
Blegdamsvej 17
DK-2100 Copenhagen Ø, Denmark

**Lance M Optican**
Laboratory of Sensorimotor Research
National Eye Institute
Bethesda MD 20892 USA

**Troels Kjær**
NORDITA
Blegdamsvej 17
DK-2100 Copenhagen Ø, Denmark

## Abstract

We have investigated the properties of neurons in inferior temporal (IT) cortex in monkeys performing a pattern matching task. Simple back-propagation networks were trained to discriminate the various stimulus conditions on the basis of the measured neuronal signal. We also trained networks to predict the neuronal response waveforms from the spatial patterns of the stimuli. The results indicate that IT neurons convey temporally encoded information about both current and remembered patterns, as well as about their behavioral context.

# 1   INTRODUCTION

Anatomical and neurophysiological studies suggest that there is a cortical pathway specialized for visual object recognition, beginning in the primary visual cortex and ending in the inferior temporal (IT) cortex (Ungerleider and Mishkin, 1982). Studies of IT neurons in awake behaving monkeys have found that visually elicited responses depend on the pattern of the stimulus and on the behavioral context of the stimulus presentation (Richmond and Sato, 1987; Miller et al, 1991). Until now, however, no attempt had been made to quantify the temporal pattern of firing in the context of a behaviorally complex task such as pattern recognition.

Our goal was to examine the information present in IT neurons about visual stimuli and their behavioral context. We explicitly allowed for the possibility that this information was encoded in the temporal pattern of the response. To decode the responses, we used simple feed-forward networks trained by back propagation.

In work reported elsewhere (Eskandar et al, 1991) this information is calculated another way, with similar results.

# 2   THE EXPERIMENT

Two monkeys were trained to perform a sequential nonmatch to sample task using a complete set of 32 black-and-white patterns based on 2-D Walsh functions. While the monkey fixated and grasped a bar, a sample pattern appeared for 352 msecs; after a pause of 500 msecs a test stimulus appeared for 352 msecs. The monkey indicated whether the test stimulus failed to match the sample stimulus by releasing the bar. (If the test matched the stimulus, the monkey waited for a third stimulus, different from the sample, before releasing the bar; see Fig. 1.)

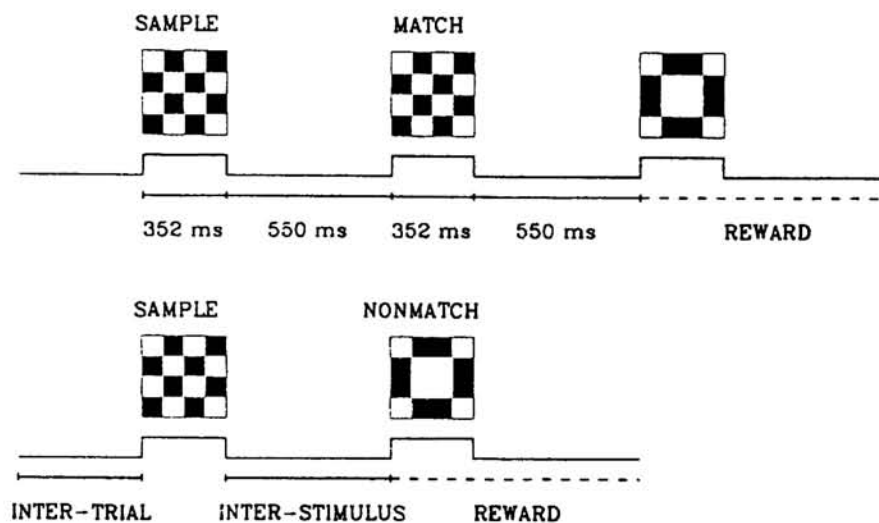

Figure 1: The nonmatch–to–sample task.

The type of trial (match or nonmatch) and the pairings of sample stimuli with nonmatch stimuli were selected randomly. A single experiment usually contained several thousand trials; thus each of the 32 patterns appeared repeatedly under the three conditions (sample, match, and nonmatch). Single neuron recordings from IT cortex were carried out while the monkeys were performing the task.

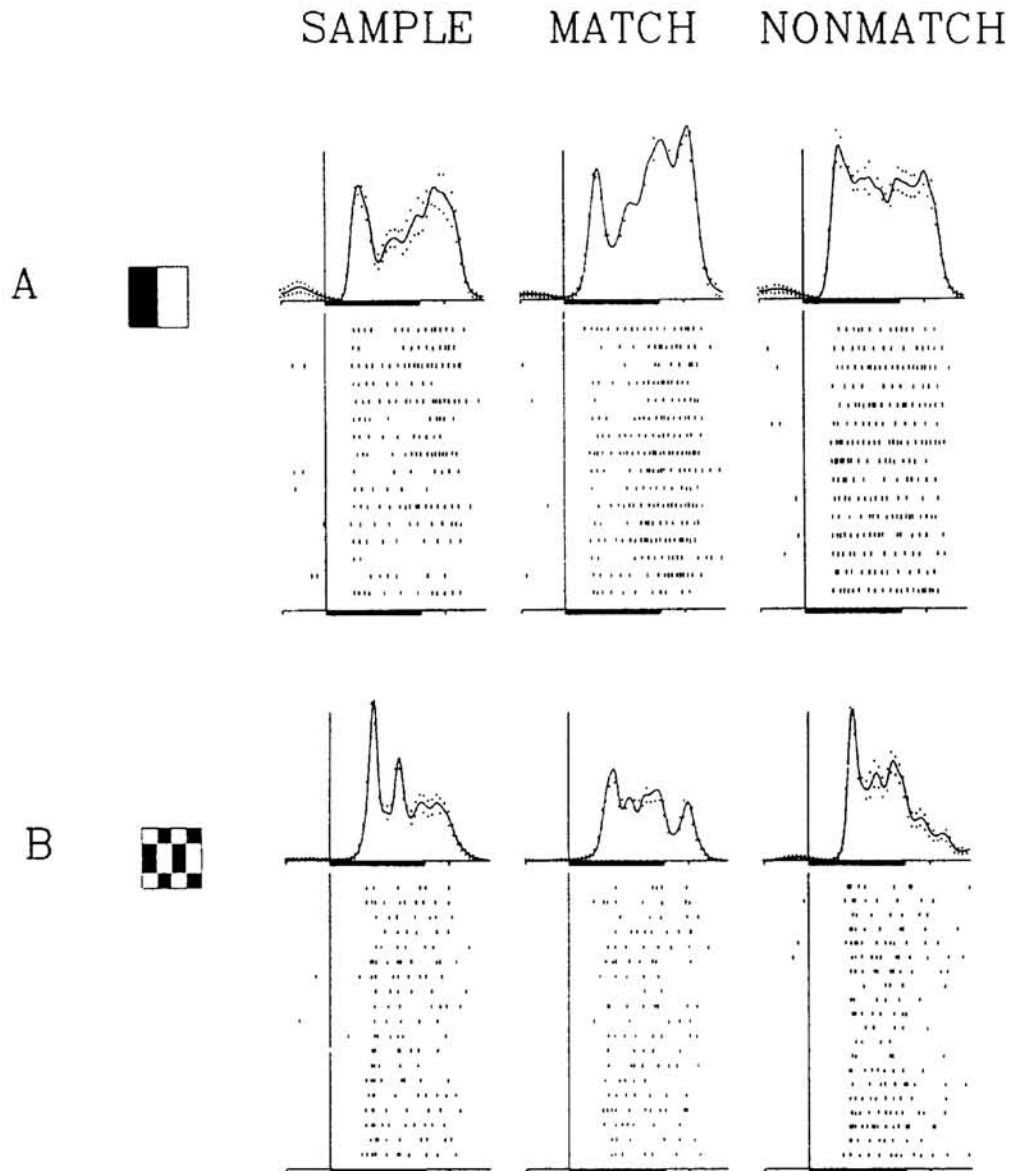

Figure 2: Responses produced by 2 stimuli under 3 behavioural conditions.

Fig. 2 shows the neuronal signals produced by two different stimulus patterns in the three behavioural conditions: sample, match and nonmatch. The lower parts of the figure show single-trial spike trains, while the upper parts show the effective time-dependent firing probabilities, inferred from the spike trains by convolving

each spike with a Gaussian kernel, adding these up for each trial and averaging the resulting continuous signals over trials. It is evident that for a given stimulus pattern the average signals produced in different behavioural conditions are different. In what follows, we proceed further to show that there is information about behavioural condition in the signal produced *in a single trial*. We will compute its average value explicitly.

## 3   DECODING NETWORKS

To compute this information we trained networks to decode the measured signal. The form of the network is shown in Fig. 3.

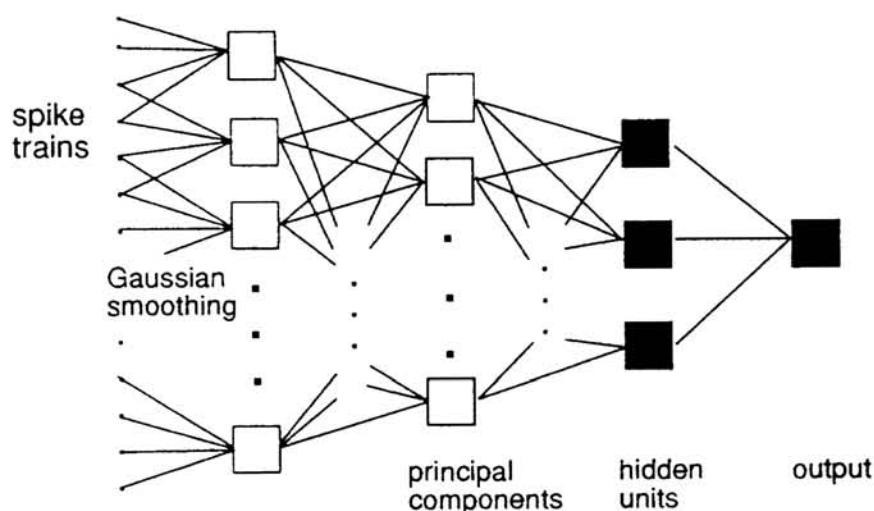

Figure 3: Network to decode neuronal signals for information about behavioural condition.

The first two layers of the network shown preprocess the spike trains as follows: We begin with the spikes measured in an interval starting 90 msec after the stimulus onset and lasting 256 msec. First each spike is convolved with a Gaussian kernel to produce a continuous signal. This signal is sampled at 4-msec intervals, giving a 64-dimensional input vector. In the second step this input vector is compressed by throwing out all but a small number of its principal components (PC's). The PC basis was obtained by diagonalizing the $64 \times 64$ covariance matrix of the inputs computed over all trials in the experiment. The remaining PC's are then the input to the rest of the network, which is a standard one with one further hidden layer. Earlier work showed that the first five PC's transmit most of the pattern information in a neuronal response (Richmond et al, 1987). Furthermore, the first PC is highly correlated with the spike count. Thus, our subsequent analysis was either on the first PC alone, as a measure of spike count, or on the first five PC's, as a measure

that incorporates temporal modulation.

We trained the networks to make pairwise discriminations between responses measured under different conditions (sample-match, sample-nonmatch, or match-nonmatch). Thus there is a single output unit, and the target is a 1 or 0 according to the behavioural condition under which that spike train was measured.

The final two layers of the network were trained by standard backpropagation of errors for the cross-entropy cost function

$$E = \sum_{\mu} \left\{ T^{\mu} \log \left[ \frac{T^{\mu}}{O(\mathbf{x}^{\mu})} \right] + (1 - T^{\mu}) \log \left[ \frac{1 - T^{\mu}}{1 - O(\mathbf{x}^{\mu})} \right] \right\}, \tag{1}$$

where $T^{\mu}$ is the target and $O^{\mu}$ the network output produced by the input vector $\mathbf{x}^{\mu}$ for training example $\mu$. The output of the network with the weights that result from this training is then the optimal estimate (given the chosen architecture) of the probability of a behavioural condition, given the measured neuronal signal used as input. The number of hidden units was adjusted to minimize the generalization error, which was computed on one quarter of the data that was reserved for this purpose.

We then calculated the mean equivocation,

$$\epsilon = -\langle O(\mathbf{x}) \log(O(\mathbf{x}) + [1 - O(\mathbf{x})] \log[1 - O(\mathbf{x})] \rangle_{\mathbf{x}}, \tag{2}$$

where $O(\mathbf{x})$ is the value of the output unit for input $\mathbf{x}$ and the average is over all inputs. (We calculated this by averagng over the test or training sets; the results were not sensitive to which one we chose.) The equivocation is a measure of the neuron's uncertainty with respect to a given discrimination. From it we can compute the transmitted information

$$I = I_{a\,priori} - \epsilon = 1 - \epsilon. \tag{3}$$

The last equality follows because in our data sets the two conditions always occur equally often.

It is evident from Fig. 2 that if we already know that our signal is produced by a particular stimulus pattern, the discrimination of the behavioural condition will be easier than if we do not possess this *a priori* knowledge. This is because the signal varies with stimulus as well as behavioural condition (more strongly, in fact), and the dependence on the latter has to be sorted out from that on the former. To get an idea of the effect of this "distraction", we performed 4 separate calculations for each of the 3 behavioural-condition discriminations, using 1, 4, 8, and all 32 stimulus patterns, respectively.

The results are summarized in Fig. 4, which shows the transmitted information about the 3 different behavioural-condition discriminations at the various levels of distraction, averaged over 5 cells. It also indicates how much of the transmitted information in each case is contained in the spike count alone (i.e. the first PC of the signal).

It is apparent that measurable information about behavioural condition is present in a single neuronal response, even in the total absence of *a priori* information about the stimulus pattern. It is also evident that most of this information is contained in

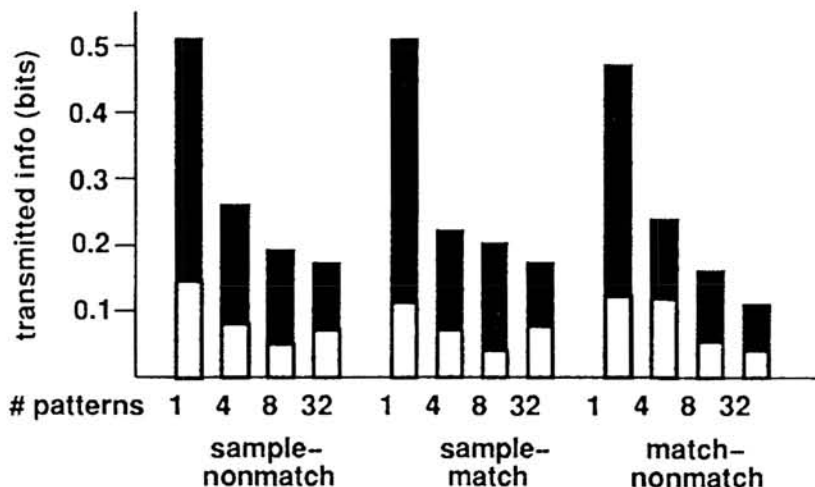

Figure 4: Transmitted information for the three behavioural discriminations with different numbers of patterns. The lower white region on each bar shows the information transmitted in the first PC alone.

the time-dependence of the firing: the information contained in the first PC of the signal is significantly less (paired t-test $p < 0.001$) and was barely out of the noise.

A finite data set can lead to a biased estimate of the transmitted information (Optican et al, 1991). In order to control for this we made a preliminary study of the dependence of the calculated equivocation on training set size. We varied the number of trials available to the network in a range (64 - 1024) for one pair of discriminations (sample vs. nonmatch). The calculated apparent equivocation increased with the sample size $N$, indicating a small-sample bias. The best correlation (Pearson $r = -0.86$) was obtained with a fit of the form:

$$\epsilon(N) = \epsilon_\infty - cN^{-1/2} \quad (c > 0). \tag{4}$$

This gives us a systematic way to estimate the small-sample bias and thus provide an improved estimate $\epsilon_\infty$ of the true equivocation. Details will be reported elsewhere.

## 4   PREDICTING NEURONAL RESPONSES

In a second set of analyses, we examined the neuronal encoding of both current and recalled patterns. The networks were trained to predict the neuronal response (as represented by its first 5 PC's) from the spatial pattern of the current nonmatch stimulus, that of the immediately preceding sample stimulus, or both. The inputs were the pixel values of the patterns.

The network is shown in Fig. 5. In order to avoid having different architectures for predictions from one and two input patterns, we always used a number of input units

equal to twice the number of pixels in the input. In the case where the prediction was to be made on the basis of both previous and current patterns, each pattern was fed into half the input units. For prediction from just one pattern (either the current or previous one), the single input pixel array was loaded separately onto both halves of the input array. As in the previous analyses, the number of hidden units was fixed by testing on a quarter of the data held out of the training set for this purpose.

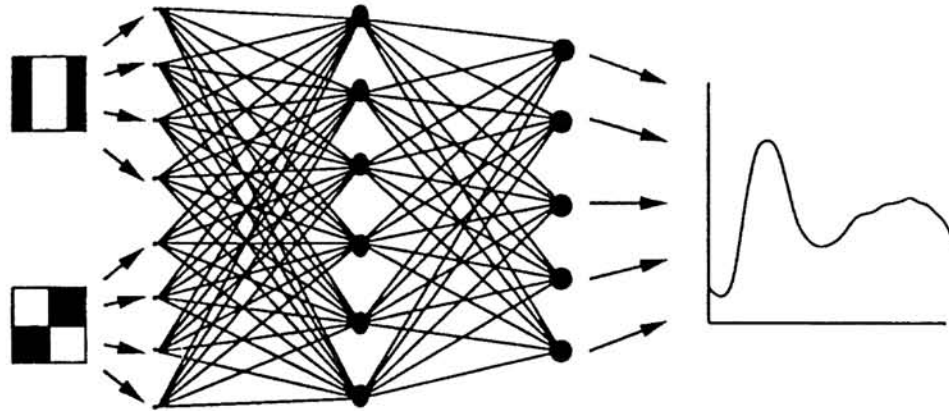

Figure 5: Network for predicting neuronal responses from the stimulus. The inputs are pixel values of the stimuli (see text), and the targets are the first 5 PC's of the measured response.

We performed this analysis on data from 6 neurons. Not surprisingly, the predicted waveforms were better when the input was the current pattern (normalized mean square error (mse) = 0.482) than when it was the previous pattern (mse = 0.589). However, the best prediction was obtained when the input reflected both the current and previous patterns (mse = 0.422). Thus the neurons we analyzed conveyed information about both remembered and current stimuli.

## 5   CONCLUSION

The results presented here demonstrate the utility of connectionist networks in analyzing neuronal information processing. We have shown that temporally modulated responses in IT cortical neurons convey information about both spatial patterns and behavioral context. The responses also convey information about the patterns of remembered stimuli. Based on these results, we hypothesize that inferior temporal neurons play a role in comparing visual patterns with those presented at an earlier time.

## Acknowledgements

This work was supported by NATO through Collaborative Research Grant CRG 900189. EE received support from the Howard Hughes Medical Institute as an NIH Research Scholar.

## References

E N Eskandar et al (1991): Inferior temporal neurons convey information about stimulus patterns and their behavioral relevance, *Soc Neurosci Abstr* **17** 443; Role of inferior temporal neurons in visual memory, submitted to *J Neurophysiol.*

E K Miller et al (1991): A neural mechanism for working and recognition memory in inferior temporal cortex, *Science* **253**

L M Optican et al (1991): Unbiased measures of transmitted information and channel capacity from multivariate neuronal data, *Biol Cybernetics* **65** 305-310.

B J Richmond and T Sato (1987): Enhancement of inferior temporal neurons during visual discrimination, *J NeurophysioL* **56** 1292-1306.

B J Richmond et al (1987): Temporal encoding of two-dimensional patterns by single units in primate inferior temporal cortex, *J Neurophysiol* **57** 132-178.

L G Ungerleider and M Mishkin (1982): Two cortical visual systems, in *Analysis of Visual Behavior*, ed. D J Ingle, M A Goodale and R J W Mansfield, pp 549-586. Cambridge: MIT Press.